# Recursive Attribute Factoring

**David Cohn**
Google Inc.,
1600 Amphitheatre Parkway
Mountain View, CA 94043
cohn@google.com

**Deepak Verma**
Dept. of CSE, Univ. of Washington,
Seattle WA- 98195-2350
deepak@cs.washington.edu

**Karl Pfleger**
Google Inc.,
1600 Amphitheatre Parkway
Mountain View, CA 94043
kpfleger@google.com

## Abstract

Clustering, or factoring of a document collection attempts to "explain" each observed document in terms of one or a small number of inferred prototypes. Prior work demonstrated that when links exist between documents in the corpus (as is the case with a collection of web pages or scientific papers), building a joint model of document contents and connections produces a better model than that built from contents or connections alone.

Many problems arise when trying to apply these joint models to corpus at the scale of the World Wide Web, however; one of these is that the sheer overhead of representing a feature space on the order of billions of dimensions becomes impractical.

We address this problem with a simple representational shift inspired by probabilistic relational models: instead of representing document linkage in terms of the *identities* of linking documents, we represent it by the explicit and inferred *attributes* of the linking documents. Several surprising results come with this shift: in addition to being computationally more tractable, the new model produces factors that more cleanly decompose the document collection. We discuss several variations on this model and show how some can be seen as exact generalizations of the PageRank algorithm.

## 1  Introduction

There is a long and successful history of decomposing collections of documents into factors or clusters to identify "similar" documents and principal themes. Collections have been factored on the basis of their textual contents [1, 2, 3], the connections between the documents [4, 5, 6], or both together [7].

A factored corpus model is usually composed of a small number of "prototype" documents along with a set of mixing coefficients (one for each document in the corpus). Each prototype corresponds to an abstract document whose features are, in some mathematical sense, "typical" of some subset of the corpus documents. The mixing coefficients for a document $\mathbf{d}$ indicate how the model's prototypes can best be combined to approximate $\mathbf{d}$.

Many useful applications arise from factored models:

- Model prototypes may be used as "topics" or cluster centers in spectral clustering [8] serving as "typical" documents for a class or cluster.

- Given a topic, factored models of link corpora allow identifying authoritative documents on that topic [4, 5, 6].

- By exploiting correlations and "projecting out" uninformative terms, the space of a factored model's mixing coefficients can provide a measure of semantic similarity between documents, regardless of the overlap in their actual terms [1].

The remainder of this paper is organized as follows: Below, we first review the vector space model, formalize the factoring problem, and describe how factoring is applied to linked document collections. In Section 2 we point out limitations of current approaches and introduce *Attribute Factoring* (AF) to address them. In the following two sections, we identify limitations of AF and describe *Recursive Attribute Factoring* and several other variations to overcome them, before summarizing our conclusions in Section 5.

**The Vector Space Model:** The vector space model is a convention for representing a document corpus (ordinarily sets of strings of arbitrary length) as a matrix, in which each document is represented as a column vector.

Let the number of documents in the corpus be $N$ and the size of vocabulary $M$. Then $\mathbf{T}$ denotes the $M \times N$ term-document matrix such that column $j$ represents document $\mathbf{d}_j$, and $\mathbf{T}_{ij}$ indicates the number of times term $t_i$ appears in document $\mathbf{d}_j$. Geometrically, the columns of $\mathbf{T}$ can also be viewed as points in an $M$ dimensional space, where each dimension $i$ indexes the number of times term $t_i$ appears in the corresponding document.

A link-based corpus may also be represented as a vector space, defining an $N \times N$ matrix $\mathbf{L}$ where $\mathbf{L}_{ij} = 1$ if there is a link from document $i$ to $j$ and 0 otherwise. It is sometimes preferable to work with $\mathbf{P}$, a normalized version of $\mathbf{L}$ in which $\mathbf{P}_{ij} = \mathbf{L}_{ij} / \sum_{i'} \mathbf{L}_{i'j}$; that is, each document's outlinks sum to 1.

**Factoring:** Let $\mathbf{A}$ represent a matrix to be factored (usually $\mathbf{T}$ or $\mathbf{T}$ augmented with some other matrix) into $K$ factors. Factoring decomposes $\mathbf{A}$ into two matrices $\mathbf{U}$ and $\mathbf{V}$ (each of rank $K$) such that $\mathbf{A} \approx \mathbf{U}\,\mathbf{V}$.[1] In the geometric interpretation, columns of $\mathbf{U}$ contains the $K$ prototypes, while columns of $\mathbf{V}$ indicate what mixture of prototypes best approximates the columns in the original matrix.

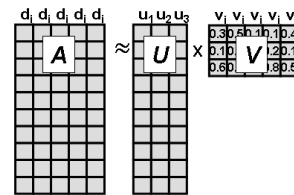

Figure 1: Factoring decomposes matrix $\mathbf{A}$ into matrices $\mathbf{U}$ and $\mathbf{V}$

The definition of what constitutes a "best approximation" leads to the many different factoring algorithms in use today. Latent Semantic Analysis [1] minimizes the sum squared reconstruction error of $\mathbf{A}$, PLSA [2] maximizes the log-likelihood that a generative model using $\mathbf{U}$ as prototypes would produce the observed $\mathbf{A}$, and Non-Negative Matrix Factorization [3] adds constraints that all components of $\mathbf{U}$ and $\mathbf{V}$ must be greater than or equal to zero.

For the purposes of this paper, however, we are agnostic as to the factorization method used — our main concern is how $\mathbf{A}$, the document matrix to be factored, is generated.

## 1.1 Factoring Text and Link Corpora

When factoring a text corpus (e.g. via LSA [1], PLSA [2], NMF [3] or some other technique), we directly factor the matrix $\mathbf{T}$. Columns of the resulting $M \times K$ matrix $\mathbf{U}$ are often interpreted as the $K$ "principal topics" of the corpus, while columns of the $K \times N$ matrix $\mathbf{V}$ are "topic memberships" of the corpus documents.

When factoring a link corpus (e.g. via ACA [4] or PHITS [6]), we factor $\mathbf{L}$ or the normalized link matrix $\mathbf{P}$. Columns of the resulting $N \times K$ matrix $\mathbf{U}$ are often interpreted as the $K$ "citation communities" of the corpus, and columns of the $K \times N$ matrix $\mathbf{V}$ indicate to what extent each document belongs to the corresponding community. Additionally, $\mathbf{U}_{ij}$, the degree of citation that community $j$ accords to document $\mathbf{d}_i$ can be interpreted as the "authority" of $\mathbf{d}_i$ in that community.

## 1.2 Factoring Text and Links Together

Many interesting corpora, such as scientific literature and the World Wide Web, contain both text content and links. Prior work [7] has demonstrated that building a single factored model of the joint term-link matrix produces a better model than that produced by using text or links alone.

The naive way to produce such a joint model is to append $\mathbf{L}$ or $\mathbf{P}$ below $\mathbf{T}$, and factor the joint matrix:

$$\begin{bmatrix} \mathbf{T} \\ \mathbf{L} \end{bmatrix} \approx \begin{bmatrix} \mathbf{U}_{\mathbf{T}} \\ \mathbf{U}_{\mathbf{L}} \end{bmatrix} \times \mathbf{V}. \tag{1}$$

When factored, the resulting $\mathbf{U}$ matrix can be seen as having two components, representing the two distinct types of information in $[\mathbf{T} ; \mathbf{L}]$. Column $i$ of $\mathbf{U}_{\mathbf{T}}$ indicates the expected term distribution of factor $i$, while the corresponding column of $\mathbf{U}_{\mathbf{L}}$ indicates the distribution of documents that typically link to documents represented by that factor.

In practice, $\mathbf{L}$ should be scaled by some factor $\lambda$ to control the relative importance of the two types of information, but empirical evidence [7] suggests that performance is somewhat insensitive to its exact value. For clarity, we omit reference to $\lambda$ in the equations below.

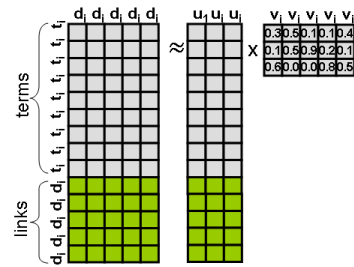

Figure 2: The naive joint model concatenates term and link matrices

## 2 Beyond the Naive Joint Model

Joint models provide a systematic way of incorporating information from both the terms and link structure present in a corpus. But the naive approach described above does not scale up to web-sized corpora, which may have millions of terms and tens of billions of documents. The matrix resulting from a naive representation of a web-scale problem would have $N + M$ features with $N \approx 10^{10}$ and $M \approx 10^6$. Simply representing this matrix (let alone factoring it) is impractical on a modern workstation.

Work on Probabilistic Relational Models (PRMs) [9] suggests another approach. The terms in a document are explicit attributes; links to the document provide additional attributes, represented (in the naive case) as the identities of the inlinking documents. In a PRM however, entities are represented by their attributes, rather than their identities. By taking a similar tack, we arrive at *Attribute Factoring* — the approach of representing link information in terms of the *attributes* of the inlinking documents, rather than by their explicit identities.

## 2.1 Attribute Factoring

Each document $\mathbf{d}_j$, along with an attribute for each term, has an attribute for each other document $\mathbf{d}_i$ in the corpus, signifying the presence (or absence) of a link from $\mathbf{d}_i$ to $\mathbf{d}_j$. When $N \approx 10^{10}$, keeping each document identity as a separate attribute is prohibitive. To create a more economical representation, we propose replacing the link attributes by a smaller set of attributes that "summarize" the information from link matrix $\mathbf{L}$, possibly in combination with the term matrix $\mathbf{T}$.

The most obvious attributes of a document are what terms it contains. Therefore, one simple way to represent the "attributes" of a document's inlinks is to aggregate the terms in the documents that link to it. There are many possible ways to aggregate these terms, including Dirichlet and more sophisticated models. For computational and representational simplicity in this paper, however, we replace inlink identities with a sum of the terms in the inlinking documents. In matrix notation, this

is just

$$\left[\begin{array}{c} \mathbf{T} \\ \mathbf{T} \times \mathbf{L} \end{array}\right] \approx \left[\begin{array}{c} \mathbf{U_T} \\ \mathbf{U_{T \times L}} \end{array}\right] \times \mathbf{V}. \qquad (2)$$

Colloquially, we can look at this representation as saying that a document has "some distribution of terms" ($\mathbf{T}$) and is linked to by documents that have "some other term distribution" ($\mathbf{T} \times \mathbf{L}$).

By substituting the aggregated attributes of the inlinks for their identities, we can reduce the size of the representation down from $(M+N) \times N$ to a much more manageable $2M \times N$. What is surprising is that, on the domains tested, this more compact representation actually *improves* factoring performance.

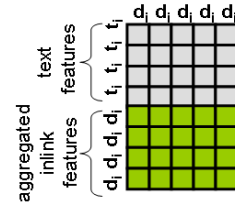

Figure 3: Representation for Attribute Factoring

## 2.2 Attribute Factoring Experiments

We tested Attribute Factoring on two publicly-available corpora of interlinked text documents. The *Cora* dataset [10] consists of abstracts and references of of approximately 34,000 computer science research papers; of these we used the approximately 2000 papers categorized into the seven subfields of machine learning. The *WebKB* dataset [11] consists of approximately 6000 web pages from computer science departments, classified by school and category (student, course, faculty, etc.).

For both datasets, we factored the content-only, naive joint, and AF joint representations using PLSA [2]. We varied $K$, the number of computed factors from 2 to 16, and performed 10 factoring runs for each value of $K$ tested. The factored models were evaluated by clustering each document to its dominant factor and measuring cluster precision: the fraction of documents in a cluster sharing the majority label.

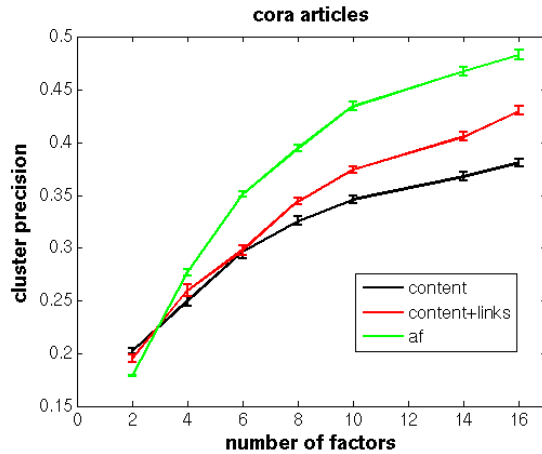

Figure 4: Attribute Factoring outperforms the content-only and naive joint representations

Figure 4 illustrates a typical result: adding explicit link information improves cluster precision, but abstracting the link information with Attribute Factoring improves it even more.

## 3 Beyond Simple Attribute Factoring

Attribute Factoring reduces the number of attributes from $N+M$ to $2M$, allowing existing factoring techniques to scale to web-sized corpora. This reduction in number of attributes however, comes at a cost. Since the identity of the document itself is replaced by its attributes, it is possible for unscrupulous authors (spammers) to "pose" as a legitimate page with high PageRank.

Consider the example shown in Figure 5, showing two subgraphs present in the web. On the right is a legitimate page like the *Yahoo!* homepage, linked to by many pages, and linking to page RYL (Real Yahoo Link). A link from the *Yahoo!* homepage to RYL imparts a lot of authority and hence is highly desired by spammers. Failing that, a spammer might try to create a counterfeit copy of the *Yahoo!* homepage, boost its PageRank by means of a "link farm", and create a link from it to his page FYL (Fake Yahoo Link).

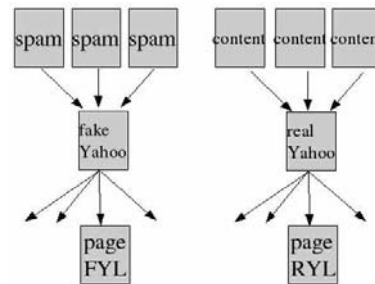

Figure 5: Attribute Factoring can be "spammed" by mirroring one level back

Without link information, our factoring can not distinguish the counterfeit homepage from the real one. Using AF or the naive joint model allows us to distinguish them based on the distribution of documents that link to each. But with AF, that real/counterfeit distinction is not propagated to documents that they point to. All that AF tells us is that RYL and FYL are pointed to by pages that look a lot like the *Yahoo!* homepage.

## 3.1 Recursive Attribute Factoring

Spamming AF was simple because it only looks one link behind. That is, attributes for a document are either explicit terms in that document or explicit terms in documents linking to the current document. This let us infer that the fake *Yahoo!* homepage was counterfeit, but provided no way to propagate this inference on to later pages.

The AF representation introduced in the previous section can be easily fooled. It makes inferences about a document based on explicit attributes propagated from the documents linking to it, but this inference only propagates one level. For example it lets us infer that the fake *Yahoo!* homepage was counterfeit, but provides no way to propagate this inference on to later pages. This suggests that we need to propagating not only *explicit* attributes of a document (its component terms), but its *inferred* attributes as well.

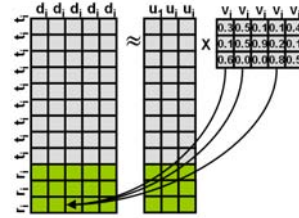

Figure 6: Recursive Attribute Factoring aggregates the inferred attributes (columns of $\mathbf{V}$) of inlinking documents

A ready source of inferred attributes comes from the factoring process itself. Recall that when factoring $\mathbf{T} \approx \mathbf{U} \times \mathbf{V}$, if we interpret the columns of $\mathbf{U}$ as factors or prototypes, then each column of $\mathbf{V}$ can be interpreted as the inferred factor memberships of its corresponding document. Therefore, we can propagate the inferred attributes of inlinking documents by aggregating the columns of $\mathbf{V}$ they correspond to (Figure 6). Numerically, this replaces $\mathbf{T}$ (the explicit document attributes) in the bottom half of the left matrix with $V$ (the inferred document attributes):

$$\left[ \begin{array}{c} \mathbf{T} \\ \mathbf{V} \times \mathbf{L} \end{array} \right] \approx \left[ \begin{array}{c} \mathbf{U_T} \\ \mathbf{U_{V \times L}} \end{array} \right] \times \mathbf{V} . \tag{3}$$

There are some worrying aspects of this representation: the document representation is no longer statically defined, and the equation itself is recursive. In practice, there is a simple iterative procedure for solving the equation (See Algorithm 1), but it is computationally expensive, and carries no convergence guarantees. The "inferred" attributes ($\mathbf{I_A}$) are set initially to random values, which are then updated until they converge. Note that we need to use the normalized version of $\mathbf{L}$, namely $\mathbf{P}$ [2].

---

**Algorithm 1** Recursive Attribute Factoring

---

1: Initialize $\mathbf{I_A}^0$ with random entries.
2: **while** Not Converged **do**
3:   Factor $\mathbf{A}^t = \left[ \begin{array}{c} \mathbf{T} \\ \mathbf{I_A}^t \end{array} \right] \approx \left[ \begin{array}{c} \mathbf{U_T} \\ \mathbf{U_{I_A}} \end{array} \right] \times \mathbf{V}$
4:   Update $\mathbf{I_A}^{t+1} = \mathbf{V} \times \mathbf{P}$ .
5: **end while**

---

## 3.2 Recursive Attribute Factoring Experiments

To evaluate RAF, we used the same data sets and procedures as in Section 2.2, with results plotted in Figure 7. It is perhaps not surprising that RAF by itself does not perform as well as AF on

the domains tested[3] - when available, explicit information is arguably more powerful than inferred information.

It's important to realize, however, that AF and RAF are in no way exclusive of each other; when we combine the two and propagate both explicit and implicit attributes, our performance is (satisfyingly) better than with either alone (top lines in Figures 7(a) and (b)).

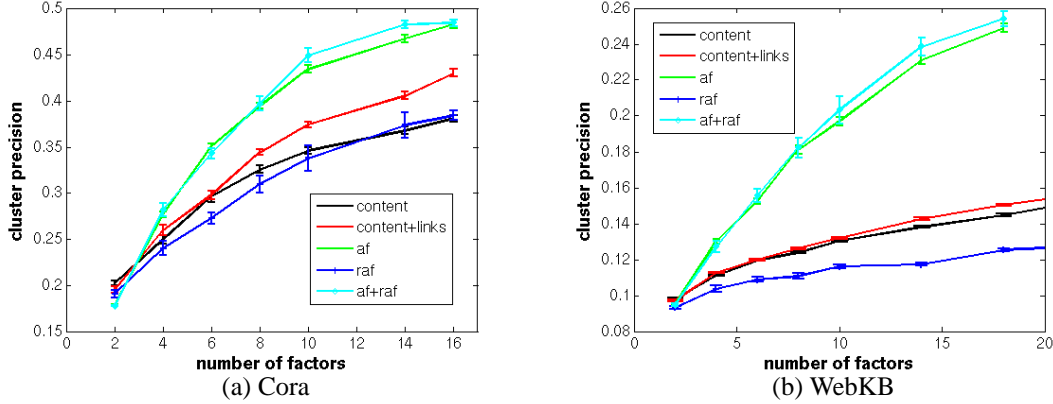

(a) Cora  (b) WebKB

Figure 7: RAF and AF+RAF results on Cora and WebKB datasets

## 4   Discussion: Other Forms of Attribute Factoring

Both Attribute Factoring and Recursive Attribute Factoring involve augmenting the term matrix with a matrix (call it $\mathbf{I_A}$) containing attributes of the inlinking documents, and then factoring the augmented matrix:

$$\mathbf{A} = \left[ \begin{array}{c} \mathbf{T} \\ \mathbf{I_A} \end{array} \right] \approx \left[ \begin{array}{c} \mathbf{U_T} \\ \mathbf{U_{I_A}} \end{array} \right] \mathbf{V}. \tag{4}$$

The traditional joint model set $\mathbf{I_A} = \mathbf{L}$; in Attribute Factoring we set $\mathbf{I_A} = \mathbf{T} \times \mathbf{L}$ and in Recursive Attribute Factoring $\mathbf{I_A} = \mathbf{V} \times \mathbf{P}$. In general though, we can set $\mathbf{I_A}$ to be any matrix that aggregates attributes of a document's inlinks.[4] For AF we can replace the $N$ dimensional inlink vector with a $M$-dimensional inferred vector $\mathbf{d}'_i$ such that $\mathbf{d}'_i = \sum_{j:\mathbf{L}_{ji}=1} w_j \mathbf{d}_j$ and then $\mathbf{I_A}$ would be the matrix with inferred attributes for each document i.e. $i^{th}$ column of $\mathbf{I_A}$ is $\mathbf{d}'_i$. Different choices for $w_j$ lead to different weighting of aggregation of attributes from the incoming documents; some variations are summarized in Table 1.

| Summed function | $w_i$ | $\mathbf{I_A}$ |
|---|---|---|
| Attribute Factoring | 1 | $\mathbf{T} \times \mathbf{L}$ |
| Outdegree-normalized Attribute Factoring | $\mathbf{P}_{ji}$ | $\mathbf{T} \times \mathbf{P}$ |
| PageRank-weighted Attribute Factoring | $\mathcal{P}_j$ | $\mathbf{T} \times \mathrm{diag}(\mathcal{P}) \times \mathbf{L}$ |
| PageRank- and outdegree-normalized | $\mathcal{P}_j \mathbf{P}_{ji}$ | $\mathbf{T} \times \mathrm{diag}(\mathcal{P}) \times \mathbf{P}$ |

Table 1: Variations on attribute weighting for Attribute Factoring. ($\mathcal{P}_j$ is PageRank of document $j$)

**Extended Attribute Factoring:**   Recursive Attribute Factoring was originally motivated by the "Fake *Yahoo!*" problem described in Section 3. While useful in conjunction with ordinary Attribute Factoring, its recursive nature and lack of convergence guarantees are troubling. One way to simulate the desired effect of RAF in a closed form is to explicitly model the inlink attributes more than just one level.[5] For example, ordinary AF looks back one level at the (explicit) attributes of inlinking documents by setting $\mathbf{I_A} = \mathbf{T} \times \mathbf{L}$. We can extend that "lookback" to two levels by defining $\mathbf{I_A} = [\mathbf{T} \times \mathbf{L}; \mathbf{T} \times \mathbf{L} \times \mathbf{L}]$. The $\mathbf{I_A}$ matrix would have $2M$ features ($M$ attributes for inlinking documents and another $M$ for attributes of documents that linked to the inlinking documents). Still, it would be possible, albeit difficult, for a determined spammer to fool this *Extended Attribute Factoring* (EAF) by mimicking two levels of the web's linkage. This can be combatted by adding a third level to the model ($\mathbf{I_A} = \left[\mathbf{T} \times \mathbf{L}; \mathbf{T} \times \mathbf{L}^2; \mathbf{T} \times \mathbf{L}^3\right]$), which increases the model complexity by only a linear factor, but (due to the web's high branching) vastly increases the number of pages a spammer would need to duplicate. It should be pointed out that these extended attributes rapidly converge to the stationary distribution of terms on the web: $\mathbf{T} \times \mathbf{L}^\infty = \mathbf{T} \times eig(\mathbf{L})$, equivalent to weighting inlinking attributes by a version of PageRank that omits random restarts. (Like in Algo. 1, $\mathbf{P}$ needs to be used instead of $\mathbf{L}$ to achieve convergence).

**Another PageRank Connection:**   While the vanilla RAF(+AF) gives good results, one can imagine many variations with interesting properties; one of them in particular is worth mentioning. A *smoothed* version of the recursive equation, can be written as

$$\left[ \begin{array}{c} \mathbf{T} \\ \epsilon + \gamma \cdot \mathbf{V} \times \mathbf{P} \end{array} \right] \approx \left[ \begin{array}{c} \mathbf{U_T} \\ \mathbf{U_{V \times L}} \end{array} \right] \times \mathbf{V}. \tag{5}$$

This the same basic equation as the RAF but multiplied with a damping factor $\gamma$. This smoothed RAF gives a further insight into working of RAF itself once we look at a simpler version of it. Starting the the original equation let us first remove the explicit attributes. This reduces the equation to $\epsilon + \gamma \cdot \mathbf{V} \times \mathbf{P} \approx \mathbf{U_{V \times L}} \times \mathbf{V}$. For the case where $U_{\mathbf{V} \times \mathbf{L}}$ has a single dimension, the above equation further simplifies to $\epsilon + \gamma \cdot \mathbf{V} \times \mathbf{P} \approx u \times \mathbf{V}$.

For some constrained values of $\epsilon$ and $\gamma$, we get $\epsilon + (1 - \epsilon) \cdot \mathbf{V} \times \mathbf{P} \approx \mathbf{V}$, which is just the equation for PageRank [12]. This means that, in the absence of $\mathbf{T}$'s term data, the inferred attributes $\mathbf{V}$ produced by smoothed RAF represent a sort of generalized, multi-dimensional PageRank, where each dimension corresponds to authority on one of the inferred topics of the corpus.[6] With the terms of $\mathbf{T}$ added, the intuition is that $\mathbf{V}$ and the inferred attributes $\mathbf{I_A} = \mathbf{V} \times \mathbf{P}$ converge to a trade-off between the generalized PageRank of link structure and factor values for $\mathbf{T}$ in terms of the prototypes $\mathbf{U_T}$ capturing term information.

# 5   Summary

We have described a representational methodology for factoring web-scale corpora, incorporating both content and link information. The main idea is to represent link information with attributes of the inlinking documents rather than their explicit identities. Preliminary results on a small dataset demonstrate that the technique not only makes the computation more tractable but also significantly improve the quality of the resulting factors.

We believe that we have only scratched the surface of this approach; many issues remain to be addressed, and undoubtedly many more remain to be discovered. We have no principled basis for weighting the different kinds of attributes in AF and EAF; while RAF seems to converge reliably in practice, we have no theoretical guarantees that it will always do so. Finally, in spite of our motivating example being the ability to factor very large corpora, we have only tested our algorithms on small "academic" data sets; applying the AF, RAF and EAF to a web-scale corpus remains as the real (and as yet untried) criterion for success.

## Footnotes

[1]In general, $A \approx f(\mathbf{U}, \mathbf{V})$, where $f$ can be any function with takes in the weights for a document and the document prototypes to generate the original vector.

[2]We would use $\mathbf{L}$ and $\mathbf{P}$ interchangeably to represent contribution from inlinking documents distinguishing only in case of "recursive" equations where it is important to normalize $\mathbf{L}$ to facilitate convergence.

[3]It is somewhat surprising (and disappointing) that RAF performs *worse* that the content-only model, but other work [7] has posited situations when this may be expected.

[4]This approach can, of course, be extended to also include attributes of the *outlinked* documents, but bibliometric analysis has historically found that inlinks are more informative about the nature of a document than outlinks (echoing the Hollywood adage that "It's not who you know that matters - it's who knows *you*").

[5]Many thanks to Daniel D. Lee for this insight.

[6]This is related to, but distinct from the generalization of PageRank described by Richardson and Domingos [13], which is computed as a scalar quantity over each of the (manually-specified) lexical topics of the corpus.

# References

[1] Scott C. Deerwester, Susan T. Dumais, Thomas K. Landauer, George W. Furnas, and Richard A. Harshman. Indexing by latent semantic analysis. *Journal of the American Society of Information Science*, 41(6):391–407, 1990.

[2] Thomas Hofmann. Probabilistic latent semantic analysis. In *Proc. of Uncertainty in Artificial Intelligence, UAI'99*, Stockholm, 1999.

[3] Daniel D. Lee and H. Sebastian Seung. Algorithms for non-negative matrix factorization. In *Advances in Neural Information Processing Systems 12*, pages 556–562. MIT Press, 2000.

[4] H.D. White and B.C. Griffith. Author cocitation: A literature measure of intellectual structure. *Journal of the American Society for Information Science*, 1981.

[5] Jon M. Kleinberg. Authoritative sources in a hyperlinked environment. *Journal of the ACM*, 46(5):604–632, 1999.

[6] David Cohn and Huan Chang. Learning to probabilistically identify authoritative documents. In *Proc. 17th International Conf. on Machine Learning*, pages 167–174. Morgan Kaufmann, San Francisco, CA, 2000.

[7] David Cohn and Thomas Hofmann. The missing link - a probabilistic model of document content and hypertext connectivity. In *Neural Information Processing Systems 13*, 2001.

[8] A. Ng, M. Jordan, and Y. Weiss. On spectral clustering: Analysis and an algorithm. In *Advances in Neural Information Processing Systems 14*, 2002.

[9] N. Friedman, L. Getoor, D. Koller, and A. Pfeffer. Learning probabilistic relational models. In *Proceedings of the Sixteenth International Joint Conference on Artificial Intelligence (IJCAI-99)*, pages 1300–1309, Stockholm, Sweden, 1999. Morgan Kaufman.

[10] Andrew K. McCallum, Kamal Nigam, Jason Rennie, and Kristie Seymore. Automating the construction of internet portals with machine learning. *Information Retrieval*, 3(2):127–163, 2000.

[11] T. Mitchell et. al. The World Wide Knowledge Base Project (Available at http://cs.cmu.edu/~WebKB). 1998.

[12] Sergey Brin and Lawrence Page. The anatomy of a large-scale hypertextual Web search engine. *Computer Networks and ISDN Systems*, 30(1–7):107–117, 1998.

[13] Mathew Richardson and Pedro Domingos. The Intelligent Surfer: Probabilistic Combination of Link and Content Information in PageRank. In *Advances in Neural Information Processing Systems 14*. MIT Press, 2002.
